# Non-parametric Regression Between Manifolds

**Florian Steinke[1], Matthias Hein[2]**
[1] Max Planck Institute for Biological Cybernetics, 72076 Tübingen, Germany
[2] Saarland University, 66041 Saarbrücken, Germany
`steinke@tuebingen.mpg.de, hein@cs.uni-sb.de`

## Abstract

This paper discusses non-parametric regression between Riemannian manifolds. This learning problem arises frequently in many application areas ranging from signal processing, computer vision, over robotics to computer graphics. We present a new algorithmic scheme for the solution of this general learning problem based on regularized empirical risk minimization. The regularization functional takes into account the geometry of input and output manifold, and we show that it implements a prior which is particularly natural. Moreover, we demonstrate that our algorithm performs well in a difficult surface registration problem.

## 1 Introduction

In machine learning, manifold structure has so far been mainly used in manifold learning [1], to enhance learning methods especially in semi-supervised learning. The setting we want to discuss in this paper is rather different, and has not been addressed yet in the machine learning community. Namely, we want to predict a mapping between *known* Riemannian manifolds based on input/output example pairs. In the statistics literature [2], this problem is treated for certain special output manifolds in directional statistics, where the main applications are to predict angles (circle), directions (sphere) or orientations (set of orthogonal matrices). More complex manifolds appear naturally in signal processing [3, 4], computer graphics [5, 6], and robotics [7]. Impressive results in shape processing have recently been obtained [8, 9] by imposing a Riemannian metric on the set of shapes, so that shape interpolation is reduced to the estimation of a smooth curve in the manifold of all shapes. Moreover, note that almost any regression problem with differentiable equality constraints can also be seen as an instance of manifold-valued learning.

The problem of learning, when input and output domain are Riemannian manifolds, is quite distinct from standard multivariate regression or manifold learning. One fundamental problem of using traditional regression methods for manifold-valued regression is that standard techniques use the linear structure of the output space. It thus makes sense to linearly combine simple basis functions, since the addition of function values is still an element of the target space. While this approach still works for manifold-valued input, it is no longer feasible if the output space is a manifold, as general Riemannian manifolds do not allow an addition operation, see Figure 1 for an illustration.

One way how one can learn manifold-valued mappings using standard regression techniques is to learn mappings directly into charts of the manifold. However, this approach leads to problems even for the simple sphere, since no *single* chart covers the sphere without a coordinate singularity. Another approach is to use an embedding of the manifold in Euclidean space where one can use standard multivariate regression and then project the learned mapping onto the manifold. But, as is obvious from Figure 1, the projection can lead to huge distortions. Even if the original mapping in Euclidean space is smooth, its projection onto the manifold might be discontinuous.

In this paper we generalize our previous work [6] which is based on regularized empirical risk minimization. The main ingredient is a smoothness functional which depends only on the geometric

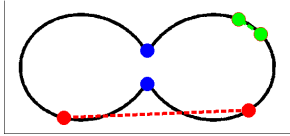
Averaging of points on the output manifold

Figure 1: The black line is a 1D-manifold in $\mathbb{R}^2$. The average of the red points in $\mathbb{R}^2$ does not lie on the manifold. Averaging of the green points which are close with respect to the geodesic distance is still reasonable. However, the blue points which are close with respect to the Euclidean distance are not necessarily close in geodesic distance and therefore averaging can fail.

properties of input and output manifold, and thus avoids the problems encountered in the naive generalization of standard regression methods discussed above. Here, we provide a theoretical analysis of the preferred mappings of the employed regularization functional, and we show that these can be seen as natural generalizations of linear mappings in Euclidean space to the manifold-valued case. It will become evident that this property makes the regularizer particularly suited as a prior for learning mappings between manifolds. Moreover, we present a new algorithm for solving the resulting optimization problem, which compared to the our previously proposed one is more robust and, most importantly, can deal with arbitrary manifold-valued input. In our implementation, the manifolds can be either given analytically or as point clouds in Euclidean space, rendering our approach applicable for almost any manifold-valued regression problem. In the experimental section we demonstrate good performance in a surface registration task, where both input manifold and output manifold are non-Euclidean – a task which could not be solved previously in [6].

Since the problem is new to the machine learning community, we give a brief summary of the learning problem in Section 2 and discuss the regularizer and its properties in Section 3. Finally, in Section 4, we describe the new algorithm for learning mappings between Riemannian manifolds, and provide performance results for a toy problem and a surface registration task in Section 5.

## 2 Regularized empirical risk minimization for manifold-valued regression

Suppose we are given two Riemannian manifolds, the input manifold $M$ of dimension $m$ and the target manifold $N$ of dimension $n$. We assume that $M$ is isometrically embedded in $\mathbb{R}^s$, and $N$ in $\mathbb{R}^t$ respectively. Since most Riemannian manifolds are given in this form anyway – think of the sphere or the set of orthogonal matrices, this is only a minor restriction. Given a set of $k$ training pairs $(X_i, Y_i)$ with $X_i \in M$ and $Y_i \in N$ we would like to learn a mapping $\Psi : M \subseteq \mathbb{R}^s \to N \subseteq \mathbb{R}^t$. This learning problem reduces to standard multivariate regression if $M$ and $N$ are both Euclidean spaces $\mathbb{R}^m$ and $\mathbb{R}^n$, and to regression on a manifold if at least $N$ is Euclidean. We use regularized empirical risk minimization, which can be formulated in our setting as

$$\underset{\Psi \in C^\infty(M,N)}{\arg\min} \frac{1}{k} \sum_{i=1}^{k} L(Y_i, \Psi(X_i)) + \lambda\, S(\Psi), \tag{1}$$

where $C^\infty(M, N)$ denotes the set of smooth mappings $\Psi$ between $M \subseteq \mathbb{R}^s$ and $N \subseteq \mathbb{R}^t$, $L : N \times N \to \mathbb{R}_+$ is the loss function, $\lambda \in \mathbb{R}_+$ the regularization parameter, and $S : C^\infty(M, N) \to \mathbb{R}_+$ the regularization functional.

**Loss function:** In multivariate regression, $f : \mathbb{R}^m \to \mathbb{R}^n$, a common loss function is the squared Euclidean distance of $f(X_i)$ and $Y_i$, $L(Y_i, f(X_i)) = \|Y_i - f(X_i)\|^2_{\mathbb{R}^n}$. A quite direct generalization to a loss function on a Riemannian manifold $N$ is to use the squared geodesic distance in $N$, $L(Y_i, \Psi(X_i)) = d_N^2(Y_i, \Psi(X_i))$. The correspondence to the multivariate case can be seen from the fact that $d_N(Y_i, \Psi(X_i))$ is the length of the shortest path between $Y_i$ and $\Psi(X_i)$ in $N$, as $\|f(X_i) - Y_i\|$ is the length of the shortest path, namely the length of the straight line, between $f(X_i)$ and $Y_i$ in $\mathbb{R}^n$.

**Regularizer:** The regularization functional should measure the smoothness of the mapping $\Psi$. We use the so-called Eells energy introduced in [6] as our smoothness functional which, as we will show in the next section, implements a particularly well-suited prior over mappings for many applications. The derivation of the regularization functional is quite technical. In order that the reader can get the main intuition without having to bother with the rather heavy machinery from differential geometry, we will discuss the regularization functional in a simplified setting, namely we assume that the input manifold $M$ is Euclidean, that is, $M$ is an open subset of $\mathbb{R}^m$. The general definition is given in the next section. Let $x^\alpha$ be Cartesian coordinates in $M$ and let $\Psi(x)$ be given in Cartesian coordinates

in $\mathbb{R}^t$ then the *Eells energy* can be written as,

$$S_{\text{Eells}}(\Psi) = \int_{M \subseteq \mathbb{R}^m} \sum_{\mu=1}^{t} \sum_{\alpha,\beta=1}^{m} \left[ \left( \frac{\partial^2 \Psi^\mu}{\partial x^\alpha \partial x^\beta} \right)^\top \right]^2 dx, \tag{2}$$

where $\top$ denotes the projection onto the tangent space $T_{\Psi(x)} N$ of the target manifold at $\Psi(x)$. Note, that the Eells energy reduces to the well-known thin-plate spline energy if also the target manifold $N$ is Euclidean, that is, $N = \mathbb{R}^n$. Let $\Psi : \mathbb{R}^m \to \mathbb{R}^n$, then

$$S_{\text{ThinPlate}}(\Psi) = \int_{M \subseteq \mathbb{R}^m} \sum_{\mu=1}^{n} \sum_{\alpha,\beta=1}^{m} \left[ \left( \frac{\partial^2 \Psi^\mu}{\partial x^\alpha \partial x^\beta} \right) \right]^2 dx. \tag{3}$$

The apparently small step of the projection onto the tangent space leads to huge qualitative differences in the behavior of both energies. In particular, the Eells energy penalizes only the second derivative *along* the manifold, whereas changes in the normal direction are discarded. In the case of $m = 1$, that is, we are learning a curve on $N$, the difference is most obvious. In this case the Eells energy penalizes only the acceleration along the curve (the change of the curve in tangent direction) whereas the thin-plate spline energy penalizes also the normal part which just measures the curvature of the curve in the ambient space. This is illustrated in the following figure.

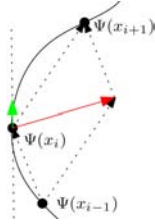
The input manifold is $\mathbb{R}$ and the output manifold $N$ is a one-dimensional curve embedded in $\mathbb{R}^2$, i.e. $\Psi : \mathbb{R} \to N$. If the images $\Psi(x_i)$ of equidistant points $x_i$ in the input manifold $M = \mathbb{R}$ are also equidistant on the output manifold, then $\Psi$ has no acceleration in terms of $N$, i.e. its second derivative in $N$ should be zero. However, the second derivative of $\Psi$ in the ambient space, which is marked red in the left figure, is not vanishing in this case. Since the manifold is curved, also the graph of $\Psi$ has to bend to stay on $N$. The Eells energy only penalizes the intrinsic acceleration, that is, only the component parallel to the tangent space at $\Psi(x_i)$, the green arrow.

## 3 Advantages and properties of the Eells energy

In the last section we motivated that the Eells energy penalizes only changes along the manifold. This property and the fact that the Eells energy is independent of the parametrization of $M$ and $N$, can be directly seen from the covariant formulation in the following section. We briefly review the derivation of the Eells energy derivation in [10], which we need in order to discuss properties of the Eells energy and the extension to manifold-valued input. Our main emphasis lies on an intuitive explanation, for the exact technical details we refer to [10].

### 3.1 The general Eells energy

Let $x^\alpha$ and $y^\mu$ be coordinates on $M$ and $N$. The differential of $\phi : M \to N$ at $x \in M$ is

$$d\phi = \frac{\partial \phi^\mu}{\partial x^\alpha} \, dx^\alpha \Big|_x \otimes \frac{\partial}{\partial y^\mu} \Big|_{\phi(x)},$$

where it is summed over double-occurring indices. This is basically just the usual Jacobian matrix for a multivariate map. In order to get a second covariant derivative of $\phi$, we apply the covariant derivative $^M\nabla$ of $M$. The problem is that the derivative $^M\nabla_{\frac{\partial}{\partial x^\alpha}} \frac{\partial}{\partial y^\mu}$ is not defined, since $\frac{\partial}{\partial y^\mu}$ is not an element of $T_x M$ but of $T_{\phi(x)} N$. For this derivative, we use the concept of the pull-back connection $\nabla'$ [11], which is given as $\nabla'_{\frac{\partial}{\partial x^\alpha}} \frac{\partial}{\partial y^\mu} = {^N\nabla}_{d\phi(\frac{\partial}{\partial x^\alpha})} \frac{\partial}{\partial y^\mu}$, i.e., the direction of differentiation $\frac{\partial}{\partial x^\alpha} \in T_x M$ is first mapped to $T_{\phi(x)} N$ using the differential $d\phi$, and then the covariant derivative $^N\nabla$ of $N$ is used. Putting things together, the second derivative, the "Hessian", of $\phi$ is given in coordinates as

$$\nabla' d\phi = \left[ \frac{\partial^2 \phi^\mu}{\partial x^\beta \partial x^\alpha} - \frac{\partial \phi^\mu}{\partial x^\gamma} {^M\Gamma}^\gamma_{\beta\alpha} + \frac{\partial \phi^\rho}{\partial x^\alpha} \frac{\partial \phi^\nu}{\partial x^\beta} {^N\Gamma}^\mu_{\nu\rho} \right] dx^\beta \otimes dx^\alpha \otimes \frac{\partial}{\partial y^\mu}, \tag{4}$$

where $^M\Gamma$, $^N\Gamma$ are the Christoffel symbols of $M$ and $N$. Note, that if $M$ and $N$ are Euclidean, the Christoffel symbols are zero and the second derivative reduces to the standard Hessian in Euclidean

space. The Eells energy penalizes the squared norm of this second derivative tensor, corresponding to the Frobenius norm of the Hessian in Euclidean space,

$$S_{\text{Eells}}(\phi) = \int_M \|\nabla' d\phi\|^2_{T_x^* M \otimes T_x^* M \otimes T_{\phi(x)} N} \, dV(x).$$

In this tensorial form the energy is parametrization independent and, since it depends only on intrinsic properties, it measures smoothness of $\phi$ only with respect to the geometric properties of $M$ and $N$. Equation (4) can be simplified significantly when $N$ is isometrically embedded in $\mathbb{R}^t$. Let $i : N \to \mathbb{R}^t$ to be the isometric embedding and denote by $\Psi : M \to \mathbb{R}^t$ the composition $\Psi = i \circ \phi$. Then we show in [10] that $\nabla' d\phi$ simplifies to

$$\nabla' d\phi = \left( \left[ \frac{\partial^2 \Psi^\mu}{\partial x^\beta \partial x^\alpha} - \frac{\partial \Psi^\mu}{\partial x^\gamma} {}^M \Gamma^\gamma_{\beta\alpha} \right] dx^\beta \otimes dx^\alpha \otimes \frac{\partial}{\partial z^\mu} \right)^\top, \tag{5}$$

where $\top$ is the orthogonal projection onto the tangent space of $N$ and $z^\mu$ are Cartesian coordinates in $\mathbb{R}^t$. Note, that if $M$ is Euclidean, the Christoffel symbols ${}^M \Gamma$ are zero and the Eells energy reduces to Equation (2) discussed in the previous section. This form of the Eells energy was also used in our previous implementation in [6] which could therefore not deal with non-Euclidean input manifolds. In this paper we generalize our setting to non-trivial input manifolds, which requires that we take into account the slightly more complicated form of $\nabla' d\phi$ in Equation (5). In Section 3.3 we discuss how to compute $\nabla' d\phi$ and thus the Eells energy for this general case.

## 3.2   The null space of the Eells energy and the generalization of linear mappings

The null space of a regularization functional $S(\phi)$ is the set $\{\phi \,|\, S(\phi) = 0\}$. This set is an important characteristic of a regularizer, since it contains all mappings which are not penalized. Thus, the null space is the set of mappings which we are free to fit the data with – only deviations from the null space are penalized. In standard regression, depending on the order of the differential used for regularization, the null space often consists out of linear maps or polynomials of small degree.

We have shown in the last section, that the Eells energy reduces to the classical thin-plate spline energy, if input and output manifold are Euclidean. For the thin-plate spline energy it is well-known that the null space consists out of linear maps between input and output space. However, the concept of linearity breaks down if the input and output spaces are Riemannian manifolds, since manifolds have no linear structure. A key observation towards a natural generalization of the concept of linearity to the manifold setting is that linear maps map straight lines to straight lines. Now, a straight line between two points in Euclidean space corresponds to a curve with no acceleration in a Riemannian manifold, that is, a geodesic between the two points. In analogy to the Euclidean case we therefore consider mappings which map geodesics to geodesics as the proper generalization of linear maps for Riemannian manifolds.

The following proposition taken from [11] defines this concept and shows that the set of generalized linear maps is exactly the null space of the Eells energy.

**Proposition 1** *[11] A map $\phi : M \to N$ is totally geodesic, if $\phi$ maps geodesics of $M$ linearly to geodesics of $N$, i.e. the image of any geodesic in $M$ is also a geodesic in $N$, though potentially with a different constant speed. We have, $\phi$ is totally geodesic if and only if $\nabla' d\phi = 0$.*

Linear maps encode a very simple relation in the data: the relative changes between input and output are the same everywhere. This is the simplest relation a non-trivial mapping can encode between input and output, and totally geodesic mappings encode the same "linear" relationship even though the input and output manifold are nonlinear. However, note that like linear maps, totally geodesic maps are *not* necessarily distortion-free, but every distortion-free (isometric) mapping is totally geodesic. Furthermore, given "isometric" training points,

$$d_M(X_i, X_j) = d_N(Y_i, Y_j), \quad i, j = 1, \dots, k,$$

then among all minimizers of (1), there will be an isometry fitting the data points, given that such an isometry exists. With this restriction in mind, one can see the Eells energy also as a measure of distortion of the mapping $\phi$. This makes the Eells energy an interesting candidate for a variety of geometric fitting problems, e.g., for surface registration as demonstrated in the experimental section.

### 3.3 Computation of the Eells energy for general input manifolds

In order to compute the Eells energy for general input manifolds, we need to be able to evaluate the second derivative in Equation (5), in particular, the Christoffel symbols of the input manifold $M$. While the Christoffel symbols could be evaluated directly for analytically given manifolds, we propose a much simpler scheme here, that also works for point clouds. It is based on local second order approximations of $M$, assuming that $M$ is given as a submanifold of $\mathbb{R}^s$ (where the Riemannian metric of $M$ is induced from the Euclidean ambient space). For simplicity, we restrict ourselves here to the intuitive case of hypersurfaces in $\mathbb{R}^s$. The case of general submanifolds in $\mathbb{R}^s$ and all proofs are provided in the supplementary material.

**Proposition 2** *Let $x^1, \ldots, x^m$ be the coordinates associated with an orthonormal basis of the tangent space at $T_pM$, that is $p$ has coordinates $x = 0$. Then in Cartesian coordinates $z$ of $\mathbb{R}^s$ centered at $p$ and aligned with the tangent space $T_pM$, the manifold can be approximated up to second order as $z(x) = (x^1, \ldots, x^{s-1}, f^s(x))$, where given that the orthonormal basis in $T_pM$ is aligned with the principal directions we have*

$$f^s(x) = \sum_{\alpha=1}^{s-1} \nu_\alpha \left(x^\alpha\right)^2,$$

*where $\nu_\alpha$ are the principal curvatures of $M$ at $x$.*

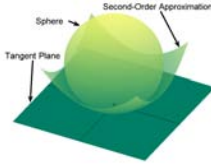

For an example of a second-order approximation, see the approximation of a sphere at the south pole on the left. Note, that the principal curvature, also called *extrinsic curvature*, quantifies how much the input manifold bends with respect to the ambient space. The principal curvatures can be computed directly for manifolds in analytic form and approximated for point cloud data using standard techniques, see Section 4.

**Proposition 3** *Given a second-order approximation of $M$ at $p$ as in Proposition 2, then for the coordinates $x$ one has $^M\Gamma^\alpha_{\beta\gamma}(0) = 0$. If $x^1, \ldots, x^{s-1}$ are aligned with the principal directions at $p$, then the coordinate expressions for the manifold-adapted second derivative of $\Psi$ (5) are at $p$*

$$\frac{\partial^2 \Psi^\mu}{\partial x^\beta \partial x^\alpha} - \frac{\partial \Psi^\mu}{\partial x^\gamma} {}^M\Gamma^\gamma_{\beta\alpha} = \frac{\partial^2 \Psi^\mu}{\partial x^\beta \partial x^\alpha} = \frac{\partial^2 \Psi^\mu}{\partial z^\beta \partial z^\alpha} + \frac{\partial \Psi^\mu}{\partial z^s} \delta_{\alpha\beta} \nu_\alpha. \tag{6}$$

Note that (6) is not an approximation, but the true second derivative of $\Psi$ at $p$ on $M$. This is because a parametrisation of $M$ at $p$ with an exponential map differs from the second order approximation at most in third order. Expression (6) allows us to compute the Eells energy in the case of manifold-valued input. We just have to replace the second-partial derivative in the Eells energy in (2) by this manifold input-adapted formulation, which can be computed easily.

## 4 Implementation

We present a new algorithm for solving the optimization problem of (1). In comparison to [6], the method is more robust, since it avoids the hard constraints of optimizing along the surface, and most importantly it allows manifold-valued input through a collocation-like discretization. The basic idea is to use a linearly parameterized set of functions and to express the objective in terms of the parameters. The resulting non-linear optimization problem is then solved using Newton's method.

**Problem Setup:** A flexible set of functions are the local polynomials. Let $M$ be an open subset or submanifold of $\mathbb{R}^s$, then we parameterize the $\mu$-th component of the mapping $\Psi : \mathbb{R}^s \to \mathbb{R}^t$ as

$$\Psi^\mu(x) = \frac{\sum_{i=1}^K k_{\sigma_i}(\|\Delta x_i\|) g(\Delta x_i, w_i^\mu)}{\sum_{j=1}^K k_{\sigma_j}(\|\Delta x_j\|)}.$$

Here, $g(\Delta x_i, w_i^\mu)$ is a first or second order polynomial in $\Delta x_i$ with parameters $w_i^\mu$, $\Delta x_i = (x - c_i)$ is the difference of $x$ to the local polynomial centers $c_i$, and $k_{\sigma_i}(r) = k(\frac{r}{\sigma_i})$ is a compactly supported smoothing kernel. We choose the $K$ local polynomial centers $c_i$ approximately uniformly distributed over $M$, thereby adapting the function class to the shape of the input manifold $M$. If we stack all parameters $w_i^\mu$ into a single vector $w$, then $\Psi$ and its partial derivatives are just linear functions of $w$,

which allows to compute these values in parallel for many points using simple matrix multiplication. We compute the energy integral (2) as a function of $w$, by summing up the energy density over an approximately uniform discretisation of $M$. The projection onto the tangent space, used in (2) and (5), and the second order approximation for computing intrinsic second derivatives, used in (5) and (6), are manifold specific and are explained below. We also express the squared geodesic distance used as loss function in terms of $w$, see below, and thus end up with a finite dimensional optimisation problem in $w$ which we solve using Newton's method with line search. The Newton step can be done efficiently because the smoothing kernel has compact support and thus the Hessian is sparse. Moreover, since we have discretised the optimisation problem directly, and not its Euler-Lagrange equations, we do not need to explicitly formulate boundary conditions.

The remaining problem is the constraint $\Psi(x) \in N$ for $x \in M$. We transform it into a soft constraint and add it to the objective function as $\gamma \int_M d(\Psi(x), N)^2 dx$, where $d(\Psi(x), N)$ denotes the distance of $\Psi(x)$ to $N$ in $\mathbb{R}^t$ and $\gamma \in \mathbb{R}_+$. During the optimization, we iteratively minimize till convergence and then increase the weight $\gamma$ by a fixed amount, repeating this until the maximum distance of $\Psi$ to $N$ is smaller than a given threshold. As initial solution, we compute the free solution, i.e. where $N$ is assumed to be $\mathbb{R}^t$, in which case the problem becomes convex. In contrast to a simple projection of the initial solution onto $N$, as done in [6], which can lead to large distortions potentially causing the optimization to stop in a local minimum, the increasing penalization of the distance to the manifold leads to a slow settling of the solution towards the target manifold, which turned out to be much more robust. The projection of the second derivative of $\Psi$ onto the tangent space for $\Psi(x) \notin N$, as required in (2) or (5), is computed using the iso-distance manifolds $N_{\Psi(x)} = \{y \in \mathbb{R}^t | d(y, N) = d(\Psi(x), N)\}$ of $N$. For the loss, we use $d_N(\mathrm{argmin}_{y \in N} \|\Psi(x) - y\|, Y_i)$. These two constructions are sensible, since as $\Psi$ approaches the manifold $N$ for increasing $\gamma$, both approximations converge to the desired operations on the manifold $N$.

**Manifold Operations:** For each output manifold $N$, we need to compute projections onto the tangent spaces of $N$ and its iso-distance manifolds, the closest point to $p \in \mathbb{R}^t$ on $N$, and geodesic distances on $N$. Using a signed distance function $\eta$, projections $P^\top$ onto the tangent spaces of $N$ or its iso-distance manifolds at $p \in \mathbb{R}^t$ are given as $P^\top = 1 - \|\nabla\eta(p)\|^{-2} \nabla\eta(p)\nabla\eta(p)^T$. For spheres $\mathbb{S}^{t-1}$ the signed distance function is simply $\eta(x) = 1 - \|x\|$. Finding the closest point to $p \in \mathbb{R}^t$ in $\mathbb{S}^{t-1}$ is trivial and the geodesic distance is $d_{\mathbb{S}^{t-1}}(x, y) = \arccos\langle x, y \rangle$ for $x, y \in \mathbb{S}^{t-1}$.

For the surface registration task, the manifold is given as a densely sampled point cloud with surface normals. Here, we proceed as follows. Given a point $p \in \mathbb{R}^t$, we first search for the closest point $p'$ in the point cloud and compute there a local second order approximation of $N$, that is, we fit the distances of the 10 nearest neighbors of $p'$ to the tangent plane (defined by the normal vector) with a quadratic polynomial in the points' tangent plane coordinates using weighted least squares, see Proposition 2. We then use the distance to the second order approximation as the desired signed distance function $\eta$, and also use this approximation to find the closest point to $p \in \mathbb{R}^t$ in $N$. Since in the surface registration problem we used rather large weights for the loss, $\Psi(X_i)$ and $Y_i$ were always close on the surface. In this case the geodesic distance can be well approximated by the Euclidean one, so that for performance reasons we used the Euclidean distance. An exact, but more expensive method to compute geodesics is to minimize the harmonic energy of curves, see [6].

For non-Euclidean input manifolds $M$, we similarly compute local second order approximations of $M$ in $\mathbb{R}^s$ to estimate the principal curvatures needed for the second derivative of $\Psi$ in (6).

## 5 Experiments

In a simple toy experiment, we show that our framework can handle noisy training data and all parameters can be adjusted using cross-validation. In the second experiment, we prove that the new implementation can deal with manifold-valued input and apply it to the task of surface registration.

**Line on Sphere:** Consider regression from $[0, 1]$ to the sphere $\mathbb{S}_2 \subseteq \mathbb{R}^3$. As ground-truth, we choose a curve given in spherical coordinates as $\phi(t) = 40t^2$, $\theta(t) = 1.3\pi t + \pi \sin(\pi t)$. The $k$ training inputs were sampled uniformly from $[0, 1]$, the outputs were perturbed by "additive" noise from the von Mises distribution with concentration parameter $\kappa$. The von Mises distribution is the maximum entropy distribution on the sphere for fixed mean and variance [2], and thus is the analog to the Gaussian distribution. In the experiments the optimal regularization parameter $\lambda$ was determined by

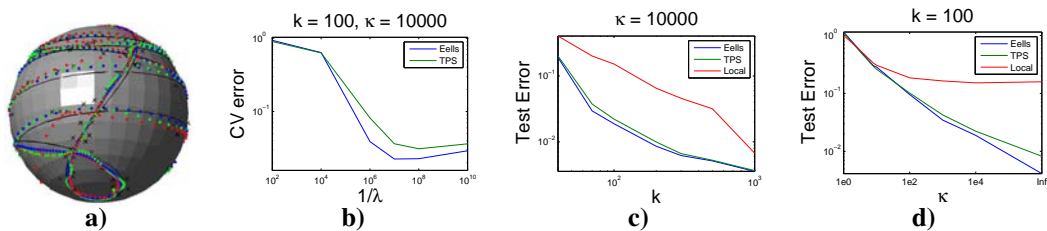

Figure 2: Regression from $[0, 1]$ to the sphere. **a)** Noisy data samples (black crosses) of the black ground-truth curve. The blue dots show the estimated curve for our Eells-regularized approach, the green dots depict thin-plate splines (TPS) in $\mathbb{R}^3$ radially projected onto the sphere, and the red dots show results for the local approach of [8]. **b)** Cross-validation errors for given sample size $k$ and noise concentration $\kappa$. Von-Mises distributed noise in this case corresponds roughly to Gaussian noise with standard deviation $0.01$. **c)** Test errors for different $k$, but fixed $\kappa$. In all experiments the regularization parameter $\lambda$ is found using cross-validation. **d)** Test errors for different $\kappa$, but fixed $k$.

performing 10-fold cross-validation and the experiment was repeated 10 times for each size of the training sample $k$ and noise parameter $\kappa$. The run-time was dominated by the number of parameters chosen for $\Psi$, and mostly independent of $k$. For training one regression it was about 10s in this case.

We compare our framework for nonparametric regression between manifolds with standard cubic smoothing splines in $\mathbb{R}^3$ – the equivalent of thin-plate splines (TPS) for one input dimension – projected radially on the sphere, and with the local manifold-valued Nadaraya-Watson estimator of [8]. As can be seen in Figure 2, our globally regularized approach performs significantly better than [8] for this task. Note that even in places where the estimated curve of [8] follows the ground truth relatively closely, the spacing between points varies greatly. These sampling dependent speed changes, that are not seen in the ground truth curve, cannot be avoided without a global smoothness prior such as for example the Eells energy. The Eells approach also outperforms the projected TPS, in particular for small sample sizes and reasonable noise levels. For a fixed noise level of $\kappa = 10000$ a paired t-test shows that the difference in test error is statistically significant at level $\alpha = 5\%$ for the sample sizes $k = 70, 200, 300, 500$. Clearly, as the curve is very densely sampled for high $k$, both approaches perform similar, since the problem becomes essentially local, and locally all manifolds are Euclidean. In contrast, for small sample sizes, a plausible a prior is more important. The necessary projections for TPS can introduce arbitrary distortions, especially for parts of the curve where consecutive training points are far apart, and where TPS thus deviate significantly from the circle, see Figure 2a). Using our manifold-adapted approach we avoid distorting projections and use the true manifold distances in the loss and the regularizer. The next example shows that the difference between TPS and our approach is even more striking for more complicated manifolds.

**Surface / Head Correspondence:** Computing correspondence between the surfaces of different, but similar objects, such as for example human heads, is a central problem in shape processing. A dense correspondence map, that is, an assignment of all points of one head to the anatomically equivalent points on the other head, allows one to perform morphing [12] or to build linear object models [13] which are flexible tools for computer graphics as well as computer vision. While the problem is well-studied, it remains a difficult problem which is still actively investigated. Most approaches minimize a functional consisting of a local similarity measure and a smoothness functional or regularizer for the overall mapping. Motivated by the fact that the Eells energy favors simple "linear" mappings, we propose to use it as regularizer for correspondence maps. For testing and highlighting the role of this "prior" independently of the choice of local similarity measure, we formulate the dense correspondence problem as a non-parametric regression problem between manifolds where 55 point correspondences on characteristic local texture or shape features are given (Only on the forehead we fix some less well-defined markers, to determine a relevant length-scale).

It is in general difficult to evaluate correspondences numerically, since for different heads anatomical equivalence is not easily specified. Here, we have used a subset of the head database of [13] and considered their correspondence as ground-truth. These correspondences are known to be perceptually highly plausible. We took the average head of one part of the database and registered it to the other 10 faces, using the mean distance to the correspondence of [13] as error score. Apart from the average deviation over the whole head, we also show results for an interior region, see Fig. 3 g), for which the correspondence given by [13] is known to be more exact compared to other regions such as, for example, around the ear or below the chin.

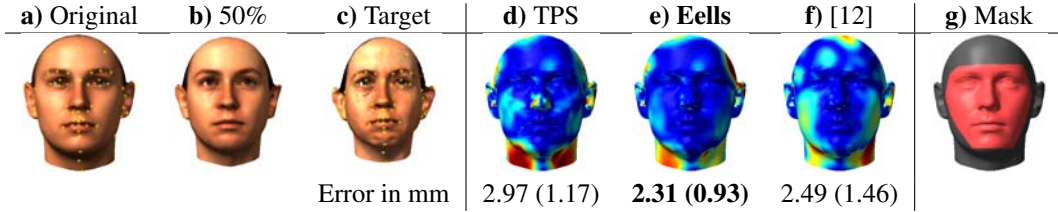

| | **a)** Original | **b)** 50% | **c)** Target | **d)** TPS | **e) Eells** | **f)** [12] | **g)** Mask |
|---|---|---|---|---|---|---|---|
| Error in mm | | | | 2.97 (1.17) | **2.31 (0.93)** | 2.49 (1.46) | |

Figure 3: Correspondence computation from the original head in **a)** to the target head in **c)** with 55 markers (yellow crosses). A resulting 50% morph using our method is shown in **c)**. Distance of the computed correspondence to the correspondence of [13] is color-coded in **d)** - **f)** for different methods. The numbers below give the average distance over the whole head, in brackets the average over an interior region (red area in **g)**.

We compared our approach against [12] and a thin-plate spline (TPS) like approach. The TPS method represents the initial solution of our approach, that is, a mapping into $\mathbb{R}^3$ minimizing the TPS energy (3), which is then projected onto the target manifold. [12] use a volume-deformation based approach that directly finds smooth mappings from surface to surface, without the need of projection, but their regularizer does not take into account the true distances along the surface. We did not compare against [8], since their approach requires computing a large number of geodesics in each iteration, which is computationally prohibitive on point clouds. In order to obtain a sufficiently flexible, yet not too high-dimensional function set for our implementation, we place polynomial centers $c_i$ on all markers points and also use a coarse, approximately uniform sampling of the other parts of the manifold. Free parameters, that is, the regularisation parameter $\lambda$ and the density of additional polynomial centers, were chosen by 10-fold cross-validation for our and the TPS method, by manual inspection for the approach of [12]. One computed correspondence example is shown in Fig. 3, the average over all 10 test heads is summarized in the table below.

| | TPS | Eells | [12] |
|---|---|---|---|
| Mean error for the full head in mm | 2.90 | 2.16 | 2.15 |
| Mean error for the interior in mm | 1.49 | 1.17 | 1.36 |

The proposed manifold-adapted Eells approach outperforms the TPS method, especially in regions of high curvature such as around the nose as the error heatmaps in Fig. 3 show. Compared to [12], our method finds a smoother, more plausible solution, also on large texture-less areas such as the forehead or the cheeks.

## References

[1] M. Belkin and P. Niyogi. Semi-supervised learning on manifolds. *Machine Learning*, 56:209–239, 2004.

[2] K.V. Mardia and P.E. Jupp. *Directional statistics*. Wiley, New York, 2000.

[3] A. Srivastava. A Bayesian approach to geometric subspace estimation. *IEEE Trans. Sig. Proc.*, 48(5):1390–1400, 2000.

[4] I. U. Rahman, I. Drori, V. C. Stodden, D. L. Donoho, and P. Schröder. Multiscale representations for manifold-valued data. *Multiscale Mod. and Sim.*, 4(4):1201–1232, 2005.

[5] F. Mémoli, G. Sapiro, and S. Osher. Solving variational problems and partial differential equations mapping into general target manifolds. *J.Comp.Phys.*, 195(1):263–292, 2004.

[6] F. Steinke, M. Hein, J. Peters, and B. Schölkopf. Manifold-valued Thin-Plate Splines with Applications in Computer Graphics. *Computer Graphics Forum*, 27(2):437–448, 2008.

[7] M. Hofer and H. Pottmann. Energy-minimizing splines in manifolds. *ACM ToG*, 23:284–293, 2004.

[8] B. C. Davis, P. T. Fletcher, E. Bullitt, and S. Joshi. Population shape regression from random design data. *Proc. of IEEE Int. Conf. Computer Vision (ICCV)*, pages 1–7, 2007.

[9] M. Kilian, N.J. Mitra, and H. Pottmann. Geometric modeling in shape space. *ACM ToG*, 26(3), 2007.

[10] M. Hein, F. Steinke, and B. Schölkopf. Energy functionals for manifold-valued mappings and their properties. Technical Report 167, MPI for Biological Cybernetics, 2008.

[11] J. Eells and L. Lemaire. *Selected topics in harmonic maps*. AMS, Providence, RI, 1983.

[12] B. Schölkopf, F. Steinke, and V. Blanz. Object correspondence as a machine learning problem. In *Proc. of the Int. Conf. on Machine Learning (ICML)*, pages 777 –784, 2005.

[13] V. Blanz and T. Vetter. A morphable model for the synthesis of 3d faces. In *SIGGRAPH'99 Conference Proceedings*, pages 187–194, Los Angeles, 1999. ACM Press.

